# Continuous Time Particle Filtering for fMRI

**Lawrence Murray**
School of Informatics
University of Edinburgh
lawrence.murray@ed.ac.uk

**Amos Storkey**
School of Informatics
University of Edinburgh
a.storkey@ed.ac.uk

## Abstract

We construct a biologically motivated stochastic differential model of the neural and hemodynamic activity underlying the observed Blood Oxygen Level Dependent (BOLD) signal in Functional Magnetic Resonance Imaging (fMRI). The model poses a difficult parameter estimation problem, both theoretically due to the nonlinearity and divergence of the differential system, and computationally due to its time and space complexity. We adapt a particle filter and smoother to the task, and discuss some of the practical approaches used to tackle the difficulties, including use of sparse matrices and parallelisation. Results demonstrate the tractability of the approach in its application to an effective connectivity study.

## 1 Introduction

Functional Magnetic Resonance Imaging (fMRI) poses a large-scale, noisy and altogether difficult problem for machine learning algorithms. The Blood Oxygen Level Dependent (BOLD) signal, from which fMR images are produced, is a measure of hemodynamic activity in the brain – only an indirect indicator of the neural processes which are of primary interest in most cases.

For studies of higher level patterns of activity, such as effective connectivity [1], it becomes necessary to strip away the hemodynamic activity to reveal the underlying neural interactions. In the first instance, this is because interactions between regions at the neural level are not necessarily evident at the hemodynamic level [2]. In the second, analyses increasingly benefit from the temporal qualities of the data, and the hemodynamic response itself is a form of temporal blurring.

We are interested in the application of machine learning techniques to reveal meaningful patterns of neural activity from fMRI. In this paper we construct a model of the processes underlying the BOLD signal that is suitable for use in a filtering framework. The model proposed is close to that of Dynamic Causal Modelling (DCM) [3]. The main innovation over these deterministic models is the incorporation of stochasticity at all levels of the system. This is important; under fixed inputs, DCM reduces to a generative model with steady state equilibrium BOLD activity and independent noise at each time point. Incorporating stochasticity allows proper statistical characterisation of the dependence between brain regions, rather than relying on relating decay rates[1].

Our work has involved applying a number of filtering techniques to estimate the parameters of the model, most notably the Unscented Kalman Filter [4] and various particle filtering techniques. This paper presents the application of a simple particle filter. [5] take a similar filtering approach, applying a local linearisation filter [6] to a model of individual regions. In contrast, the approach here is applied to multiple regions and their interactions, not single regions in isolation.

Other approaches to this type of problem are worth noting. Perhaps the most commonly used technique to date is Structural Equation Modelling (SEM) [7; 8] (e.g. [9; 10; 11]). SEM is a multivariate

regression technique where each dependent variable may be a linear combination of both independent and other dependent variables. Its major limitation is that it is static, assuming that all observations are temporally independent and that interactions are immediate and wholly evident within each single observation. Furthermore, it does not distinguish between neural and hemodynamic activity, and in essence identifies interactions only at the hemodynamic level.

The major contributions of this paper are establishing a stochastic model of latent neural and hemodynamic activity, formulating a filtering and smoothing approach for inference in this model, and overcoming the basic practical difficulties associated with this. The estimated neural activity relates to the domain problem and is temporally consistent with the stimulus. The approach is also able to establish connectivity relationships.

The ability of this model to establish such connectivity relationships on the basis of stochastic temporal relationships is significant. One problem in using structural equation models for effective connectivity analysis is the statistical equivalence of different causal models. By presuming a temporal causal order, temporal models of this form have no such equivalence problems. Any small amount of temporal connectivity information available in fMRI data is of significant benefit, as it can disambiguate between statically equivalent models.

Section 2 outlines the basis of the hemodynamic model that is used. This is combined with neural, input and measurement models in Section 3 to give the full framework. Inference and parameter estimation are discussed in Section 4, before experiments and analysis in Sections 5 and 6.

## 2   Hemodynamics

Temporal analysis of fMRI is significantly confounded by the fact that it does not measure brain activity directly, but instead via hemodynamic activity, which (crudely) temporally smooths the activity signal. The quality of temporal analysis therefore depends significantly on the quality of model used to relate neural and hemodynamic activity.

This relationship may be described using the now well established Balloon model [12]. This models a venous compartment as a balloon using Windkessel dynamics. The state of the compartment is represented by its blood volume normalised to the volume at rest, $v = V/V_0$ (blood volume $V$, rest volume $V_0$), and deoxyhemoglobin (dHb) content normalised to the content at rest, $q = Q/Q_0$ (dHb content $Q$, rest content $Q_0$). The compartment receives inflow of fully oxygenated arterial blood $f_{in}(t)$, extracts oxygen from the blood, and expels partially deoxygenated blood $f_{out}(t)$. The full dynamics may be represented by the differential system:

$$\frac{dq}{dt} = \frac{1}{\tau_0}\left[f_{in}(t)\frac{E(t)}{E_0} - f_{out}(v)\frac{q}{v}\right] \tag{1}$$

$$\frac{dv}{dt} = \frac{1}{\tau_0}\left[f_{in}(t) - f_{out}(v)\right] \tag{2}$$

$$E(t) \approx 1 - (1 - E_0)^{\frac{1}{f_{in}(t)}} \tag{3}$$

$$f_{out}(v) \approx v^{\frac{1}{\alpha}} \tag{4}$$

where $\tau_0$ and $\alpha$ are constants, and $E_0$ is the oxygen extraction fraction at rest.

This base model is driven by the independent input $f_{in}(t)$. It may be further extended to couple in neural activity $z(t)$ via an abstract vasodilatory signal $s$ [13]:

$$\frac{df}{dt} = s \tag{5}$$

$$\frac{ds}{dt} = \epsilon z(t) - \frac{s}{\tau_s} - \frac{(f-1)}{\tau_f}. \tag{6}$$

The complete system defined by Equations 1-6, with $f_{in}(t) = f$, is now driven by the independent input $z(t)$. From the balloon model, the relative BOLD signal change over the baseline $S$ at any time may be predicted using [12]:

$$\frac{\Delta S}{S} = V_0\left[k_1(1-q) + k_2\left(1 - \frac{q}{v}\right) + k_3(1-v)\right]. \tag{7}$$

Figure 1 illustrates the system dynamics. Nominal values for constants are given in Table 1.

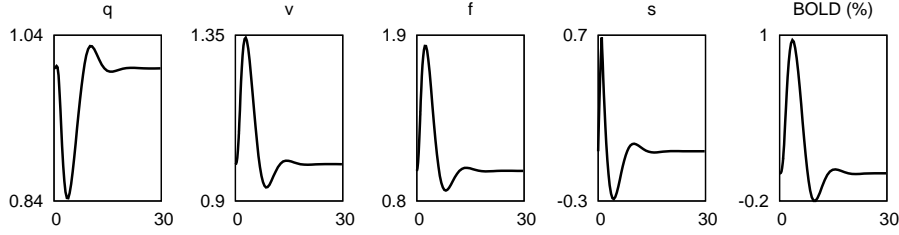

Figure 1: Response of the balloon model to a 1s burst of neural activity at magnitude 1 (time on $x$ axis, response on $y$ axis).

## 3   Model

We define a model of the neural and hemodynamic interactions between $M$ regions of interest. A region consists of neural tissue and a venous compartment. The state $\mathbf{x}_i(t)$ of region $i$ at time $t$ is given by:

$$\mathbf{x}_i(t) = \begin{cases} z_i(t) & \text{neural activity} \\ f_i(t) & \text{normalised blood flow into the venous compartment} \\ s_i(t) & \text{vasodilatory signal} \\ q_i(t) & \text{normalised dHb content of the venous compartment} \\ v_i(t) & \text{normalised blood volume of the venous compartment} \end{cases}$$

The complete state at time $t$ is given by $\mathbf{x}(t) = (\mathbf{x}_1(t)^T, \ldots, \mathbf{x}_M(t)^T)^T$.

We construct a model of the interactions between regions in four parts – the input model, the neural model, the hemodynamic model and the measurement model.

### 3.1   Input model

The input model represents the stimulus associated with the experimental task during an fMRI session. In general this is a function $\mathbf{u}(t)$ with $U$ dimensions. For a simple block design paradigm a one-dimensional box-car function is sufficient.

### 3.2   Neural model

Neural interactions between the regions are given by:

$$d\mathbf{z} = A\mathbf{z}\,dt + C\mathbf{u}\,dt + \mathbf{c} + \Sigma_{\mathbf{z}}\,d\mathbf{W}, \tag{8}$$

where $d\mathbf{W}$ is the $M$-dimensional standard (zero mean, unit variance) Wiener process, $A$ an $M \times M$ matrix of efficacies between regions, $C$ an $M \times U$ matrix of efficacies between inputs and regions, $\mathbf{c}$ an $M$-dimensional vector of constant terms and $\Sigma_{\mathbf{z}}$ an $M \times M$ diagonal diffusion matrix with $\sigma_{z_1}, \ldots, \sigma_{z_M}$ along the diagonal.

This is similar to the deterministic neural model of DCM expressed as a stochastic differential equation, but excludes the bilinear components allowing modulation of connections between seeds. In theory these can be added, we simply limit ourselves to a simpler model for this early work. In addition, and unlike DCM, nonlinear interactions between regions could also be included to account for modulatory activity. Again it seems sensible to keep the simplest linear case at this stage of the work, but the potential for nonlinear generalisation is one of the longer term benefits of this approach.

### 3.3   Hemodynamic model

Within each region, the variables $f_i$, $s_i$, $q_i$, $v_i$ and $z_i$ interact according to a stochastic extension of the balloon model (c.f. Equations 1-6). It is assumed that regions are sufficiently separate that their

| Constant | $\tau_0$ | $\tau_f$ | $\tau_s$ | $\alpha$ | $\epsilon$ | $V_0$ | $E_0$ | $k_1$ | $k_2$ | $k_3$ |
|---|---|---|---|---|---|---|---|---|---|---|
| Value | 0.98 | $1/0.65$ | $1/0.41$ | 0.32 | 0.8 | 0.018 | 0.4 | $7E_0$ | 2 | $2E_0 - 0.2$ |

Table 1: Nominal values for constants of the balloon model [12; 13].

hemodynamic activity is independent given neural activity[14]. Noise in the form of the Wiener process is introduced to $s_i$ and the log space of $f_i$, $q_i$ and $v_i$, in the latter three cases to ensure positivity:

$$d \ln f_i = \frac{1}{f_i} s_i \, dt + \sigma_{f_i} \, dW \tag{9}$$

$$ds_i = \left[ \epsilon z_i - \frac{s}{\tau_s} - \frac{(f-1)}{\tau_f} \right] dt + \sigma_{s_i} \, dW \tag{10}$$

$$d \ln q_i = \frac{1}{q_i \tau_0} \left[ f_i \frac{1 - (1 - E_0)^{\frac{1}{f_i}}}{E_0} - v_i^{\frac{1}{\alpha} - 1} q_i \right] dt + \sigma_{q_i} \, dW \tag{11}$$

$$d \ln v_i = \frac{1}{v_i \tau_0} \left[ f_i - v_i^{\frac{1}{\alpha}} \right] dt + \sigma_{v_i} \, dW. \tag{12}$$

### 3.4 Measurement model

The relative BOLD signal change at any time for a particular region is given by (c.f. Equation 7):

$$\Delta y_i = V_0 \left[ k_1 (1 - q_i) + k_2 \left( 1 - \frac{q_i}{v_i} \right) + k_3 (1 - v_i) \right]. \tag{13}$$

This may be converted to an absolute measurement $y_i^*$ for comparison with actual observations by using the baseline signal $b_i$ for each seed and an independent noise source $\xi \sim \mathcal{N}(0, 1)$:

$$y_i^* = b_i (1 + \Delta y_i) + \sigma_{y_i} \xi. \tag{14}$$

## 4 Estimation

The model is completely defined by Equations 8 to 14. This fits nicely into a filtering framework, whereby the input, neural and hemodynamic models define state transitions, and the measurement model predicted observations. For $i = 1, \ldots, M$, $\sigma_{z_i}$, $\sigma_{f_i}$, $\sigma_{s_i}$, $\sigma_{q_i}$ and $\sigma_{v_i}$ define the system noise and $\sigma_{y_i}$ the measurement noise. Parameters to estimate are the elements of $A$, $C$, $\mathbf{c}$ and $\mathbf{b}$.

For a sequence of time points $t_1, \ldots, t_T$, we are given observations $\mathbf{y}(t_1), \ldots, \mathbf{y}(t_T)$, where $\mathbf{y}(t) = (y_1(t), \ldots, y_M(t))^T$. We seek to exploit the data as much as possible by estimating $P(\mathbf{x}(t_n) \,|\, \mathbf{y}(t_1), \ldots, \mathbf{y}(t_T))$ for $n = 1, \ldots, T$ – the distribution over the state at each time point given all the data.

Because of non-Gaussianity and nonlinearity of the transitions and measurements, a two-pass particle filter is proposed to solve the problem. The forward pass is performed using a sequential importance resampling technique similar to CONDENSATION [15], obtaining $P(\mathbf{x}(t_n) \,|\, \mathbf{y}(t_1), \ldots, \mathbf{y}(t_n))$ for $n = 1, \ldots, T$. Resampling at each step is handled using a deterministic resampling method [16]. The transition of particles through the differential system uses a 4th/5th order Runge-Kutta-Fehlberg method, the adaptive step size maintaining fixed error bounds.

The backwards pass is substantially more difficult. Naively, we can simply negate the derivatives of the differential system and step backwards to obtain $P(\mathbf{x}(t_n) \,|\, \mathbf{y}(t_{n+1}), \ldots, \mathbf{y}(t_T))$, then fuse these with the results of the forwards pass to obtain the desired posterior. Unfortunately, such a backwards model is divergent in $\mathbf{q}$ and $\mathbf{v}$, so that the accumulated numerical errors of the Runge-Kutta can easily cause an explosion to implausible values and a tip-toe adaptive step size to maintain error bounds. This can be mitigated by tightening the error bounds, but the task becomes computationally prohibitive well before the system is tamed.

An alternative is a two-pass smoother that reuses particles from the forwards pass [17], reweighting them on the backwards pass so that no explicit backwards dynamics are required. This sidesteps the divergence issue completely, but is computationally and spatially expensive and requires computation of $p(\mathbf{x}(t_n) = \mathbf{s}_{t_n}^{(i)} \,|\, \mathbf{x}(t_{n-1}) = \mathbf{s}_{t_{n-1}}^{(j)})$ for particular particles $\mathbf{s}_{t_n}^{(i)}$ and $\mathbf{s}_{t_{n-1}}^{(j)}$. This imposes some limitations, but is nevertheless the method used here.

The forwards pass provides a weighted sample set $\{(\mathbf{s}_t^{(i)}, \pi_t^{(i)})\}$ at each time point $t = t_1, \ldots, t_T$ for $i = 1, \ldots, P$. Initialising with $\psi_{t_T} = \pi_{t_T}$, the backwards step to calculate weights at time $t_n$ is

as follows [17][2]:

$$
\begin{aligned}
\alpha_{t_n}^{(i,j)} &= p(\mathbf{x}(t_{n+1}) = \mathbf{s}_{t_{n+1}}^{(i)} \,|\, \mathbf{x}(t_n) = \mathbf{s}_{t_n}^{(j)}) \text{ for } i, j = 1, \dots, P \\
\gamma_{t_n} &= \alpha_{t_n} \pi_{t_n} \\
\delta_{t_n} &= \alpha_{t_n}^T (\psi_{t_{n+1}} \oslash \gamma_{t_n}) \text{ where } \oslash \text{ is element-wise division,} \\
\psi_{t_n} &= \pi_{t_n} \otimes \delta_{t_n} \text{ where } \otimes \text{ is element-wise multiplication.}
\end{aligned}
$$

These are then normalised so that $\sum \psi_{t_n}^{(i)} = 1$ and the smoothed result $\{(\mathbf{s}_{t_n}^{(i)}, \psi_{t_n}^{(i)})\}$ for $i = 1, \dots, P$ is stored.

There are numerous means of propagating particles through the forwards pass that accommodate the resampling step and propagation of the Wiener noise through the nonlinearity. These include various stochastic Runge-Kutta methods, the Unscented Transformation [4] or a simple Euler scheme using fixed time steps and adding an appropriate portion of noise after each step. The requirement to efficiently make $P^2$ density calculations of $p(\mathbf{x}(t_{n+1}) = \mathbf{s}_{t_{n+1}}^{(i)} \,|\, \mathbf{x}(t_n) = \mathbf{s}_{t_n}^{(j)})$ during the backwards pass is challenging with such approaches, however. To keep things simple, we instead simply propagate particles noiselessly through the transition function, and add noise from the Wiener process only at times $t_1, \dots, t_T$ as if the transition were linear. This reasonably approximates the noise of the system while keeping the density calculations very simple – transition $\mathbf{s}_{t_n}^{(j)}$ noiselessly to obtain the mean value of a Gaussian with covariance equal to that of the system noise, then calculate the density of this Gaussian at $\mathbf{s}_{t_{n+1}}^{(i)}$.

Observe that if system noise is sufficiently tight, $\alpha_{t_n}$ becomes sparse as negligibly small densities round to zero. Implementing $\alpha_{t_n}$ as a sparse matrix can provide significant time and space savings.

Propagation of particles through the transition function and density calculations can be performed in parallel. This applies during both passes. For the backwards pass, each particle at $t_n$ need only be transitioned once to produce a Gaussian from which the density of all particles at $t_{n+1}$ can be calculated, filling in one column of $\alpha_{t_n}$.

Finally, the parameters $A$, $C$, $\mathbf{c}$ and $\mathbf{b}$ may be estimated by adding them to the state with artificial dynamics (c.f. [18]), applying a broad prior and small system noise to suggest that they are generally constant. The same applies to parameters of the balloon model, which may be included to allow variation in the hemodynamic response across the brain.

## 5    Experiments

We apply the model to data collected during a simple finger tapping exercise. Using a Siemens Vision at 2T with a TR of 4.1s, a healthy 23-year-old right-handed male was scanned on 33 separate days over a period of two months. In each session, 80 whole volumes were taken, with the first two discarded to account for T1 saturation effects. The experimental paradigm consists of alternating 6TR blocks of rest and tapping of the right index finger at 1.5Hz, where tapping frequency is provided by a constant audio cue, present during both rest and tapping phases.

All scans across all sessions were realigned using SPM5 [19] and a two-level random effects analysis performed, from which 13 voxels were selected to represent regions of interest. No smoothing or normalisation was applied to the data. Of the 13 voxels, four are selected for use in this experiment – located in the left posterior parietal cortex, left M1, left S1 and left premotor cortex. The mean of all sessions is used as the measurement $\mathbf{y}(t)$, which consists of $M = 4$ elements, one for each region.

We set $t_1 = 1\text{TR} = 4.1\text{s}, \dots, t_T = 78\text{TR} = 319.8\text{s}$ as the sequence of times, corresponding to the times at which measurements are taken after realignment. The experimental input function $\mathbf{u}(t)$ is plotted in Figure 2, taking a value of 0 at rest and 1 during tapping. The error bounds on the Runge-Kutta are set to $10^{-4}$. Measurement noise is set to $\sigma_{y_i} = 2$ for $i = 1, \dots, M$ and the prior and system noise as in Table 2. With the elements of $A$, $C$, $\mathbf{c}$ and $\mathbf{b}$ included in the state, the state size is 48. $P = 10^6$ particles are used for the forwards pass, downsampling to $2.5 \times 10^4$ particles for the more expensive backwards pass.

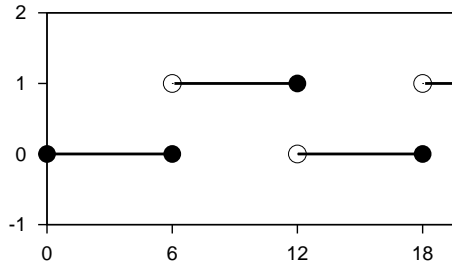

Figure 2: Experimental input $\mathbf{u}(t)$, $x$ axis is time $t$ expressed in TRs.

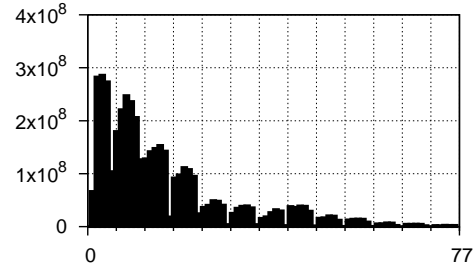

Figure 3: Number of nonzero elements in $\alpha_{t_n}$ for $n = 1, \ldots, 77$.

| | | Prior | | Noise |
|---|---|---|---|---|
| | | $\mu$ | $\sigma$ | $\sigma$ |
| $A_{i,i}$ | $i = 1, \ldots, N$ | $-1$ | $1/2$ | $10^{-2}$ |
| $A_{i,j}$ | $i, j = 1, \ldots, N, i \neq j$ | $0$ | $1/2$ | $10^{-2}$ |
| $C_{i,1}$ | $i = 1, \ldots, N$ | $0$ | $1/2$ | $10^{-2}$ |
| $z_i$ | $i = 1, \ldots, N$ | $0$ | $1/2$ | $10^{-1}$ |
| $f_i, s_i, q_i, v_i, c_i$ | $i = 1, \ldots, N$ | $0$ | $1/2$ | $10^{-2}$ |
| $b_i$ | $i = 1, \ldots, N$ | $\bar{y}_i$ | $10$ | $10^{-2}$ |

Table 2: Prior and system noise.

The experiment is run on the *Eddie* cluster of the Edinburgh Compute and Data Facility (ECDF) [3] over 200 nodes, taking approximately 10 minutes real time. The particle filter and smoother are distributed across nodes and run in parallel using the dysii Dynamic Systems Library [4].

After application of the filter, the predicted neural activity is given in Figure 4 and parameter estimates in Figures 6 and 7. The predicted output obtained from the model is in Figure 5, where it is compared to actual measurements acquired during the experiment to assess model fit.

## 6 Discussion

The model captures the expected underlying form for neural activity, with all regions distinctly correlated with the experimental stimulus. Parameter estimates are generally constant throughout the length of the experiment and some efficacies are significant enough in magnitude to provide biological insight. The parameters found typically match those expected for this form of finger tapping task. However, as the focus of this paper is the development of the filtering approach we will reserve a real analysis of the results for a future paper, and focus on the issues surrounding the filter and its capabilities and deficiencies. A number of points are worth making in this regard.

Particles stored during the forwards pass do not necessarily support the distributions obtained during the backwards pass. This is particularly obvious towards the extreme left of Figure 4, where the smoothed results appear to become erratic, essentially due to degeneracy in the backwards pass. Furthermore, while the smooth weighting of particles in the forwards pass is informative, that of the backwards pass is often not, potentially relying on heavy weighting of outlying particles and shedding little light on the actual nature of the distributions involved.

Figure 3 provides empirical results as to the sparseness of $\alpha_{t_n}$. At worst at least 25% of elements are zero, demonstrating the advantages of a sparse matrix implementation in this case.

The particle filter is able to establish consistent neural activity and parameter estimates across runs. These estimates also come with distributions in the form of weighted sample sets which enable the uncertainty of the estimates to be understood. This certainly shows the stochastic model and particle filter to be a promising approach for systematic connectivity analysis.

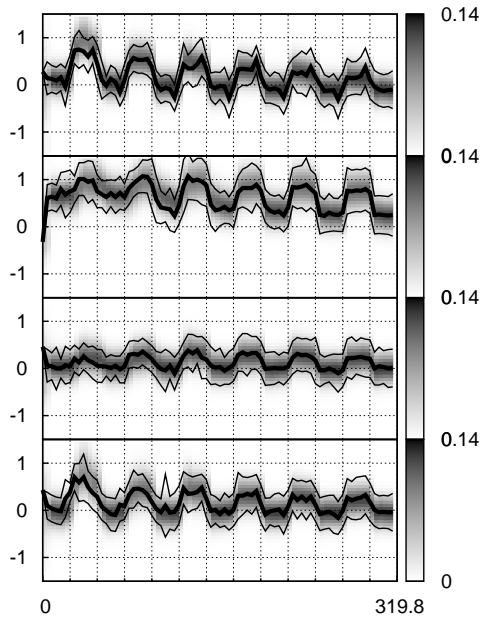

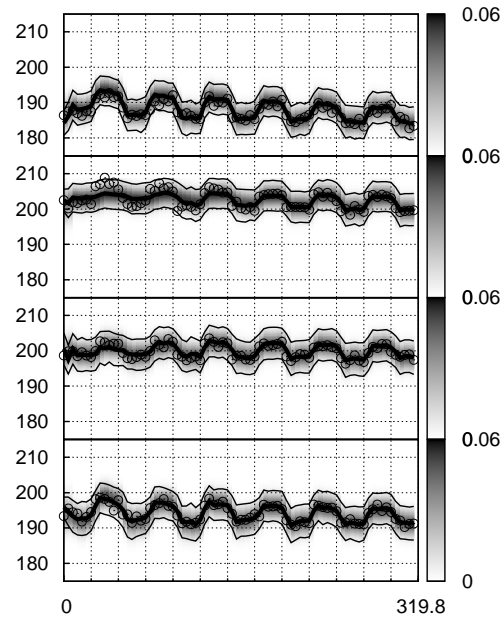

Figure 4: Neural activity predictions $\mathbf{z}$ ($y$ axis) over time ($x$ axis). Forwards pass results as shaded histogram, smoothed results as solid line with $2\sigma$ error.

Figure 5: Measurement predictions $\mathbf{y}^*$ ($y$ axis) over time ($x$ axis). Forwards pass results as shaded histogram, smoothed results as solid line with $2\sigma$ error, circles actual measurements.

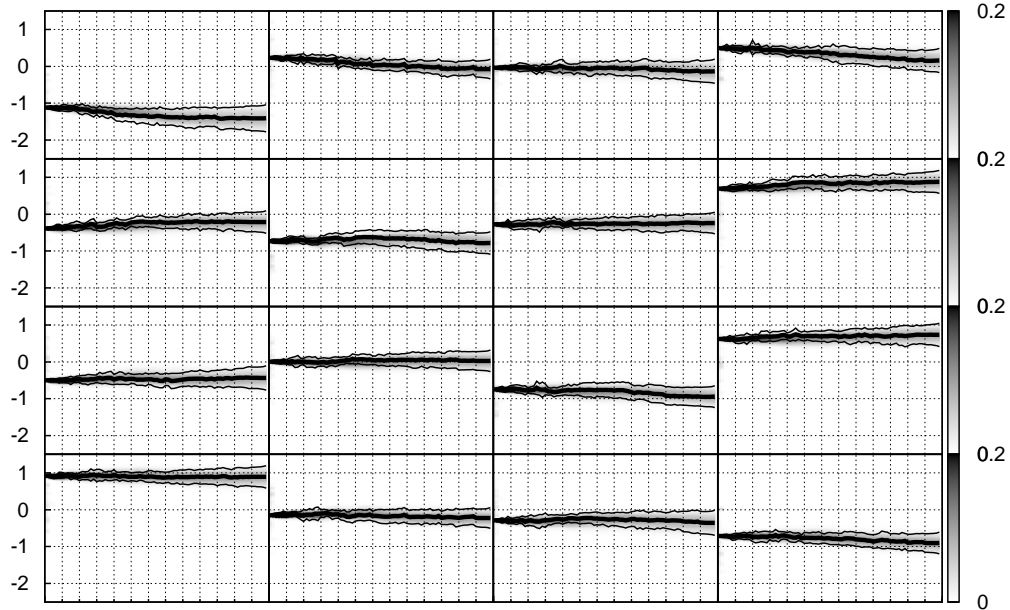

Figure 6: Parameter estimates $A$ ($y$ axis) over time ($x$ axis). Forwards pass results as shaded histogram, smoothed results as solid line with $2\sigma$ error.

The authors would like to thank David McGonigle for helpful discussions and detailed information regarding the data set.

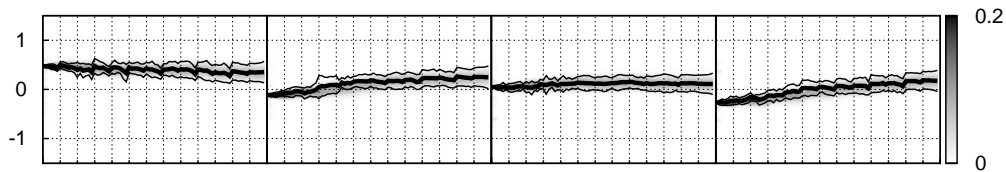

Figure 7: Parameter estimates of $C$ ($y$ axis) over time ($x$ axis). Forwards pass results as shaded histogram, smoothed results as solid line with $2\sigma$ error.

## Footnotes

[1]A good analogy is the fundamental difference between modelling time series data $y_t$ using an exponentially decaying curve with observational noise $x_t = ax_{t-1} + c$, $y_t = x_t + \epsilon_t$, and using a much more flexible Kalman filter $x_t = ax_{t-1} + c + \omega_t$, $y_t = x_t + \epsilon_t$ (where $x_t$ is a latent variable, $a$ a decay constant, $c$ a constant and $\epsilon$ and $\omega$ Gaussian variables).

[2]We have expressed this in matrix notation rather than the original notation in [17]

[3]http://www.is.ed.ac.uk/ecdf/

[4]http://www.indii.org/software/dysii/

# References

[1] Friston, K. and Buchel, C. (2004) *Human Brain Function*, chap. 49, pp. 999–1018. Elsevier.

[2] Gitelman, D. R., Penny, W. D., Ashburner, J., and Friston, K. J. (2003) Modeling regional and psychophysiologic interactions in fMRI: the importance of hemodynamic deconvolution. *NeuroImage*, **19**, 200–207.

[3] Friston, K., Harrison, L., and Penny, W. (2003) Dynamic causal modelling. *NeuroImage*, **19**, 1273–1302.

[4] Julier, S. J. and Uhlmann, J. K. (1997) A new extension of the Kalman filter to nonlinear systems. *The Proceedings of AeroSense: The 11th International Symposium on Aerospace/Defense Sensing, Simulation and Controls, Multi Sensor Fusion, Tracking and Resource Management*.

[5] Riera, J. J., Watanabe, J., Kazuki, I., Naoki, M., Aubert, E., Ozaki, T., and Kawashim, R. (2004) A state-space model of the hemodynamic approach: nonlinear filtering of BOLD signals. *NeuroImage*, **21**, 547–567.

[6] Ozaki, T. (1993) A local linearization approach to nonlinear filtering. *International Journal on Control*, **57**, 75–96.

[7] Bentler, P. M. and Weeks, D. G. (1980) Linear structural equations with latent variables. *Psychometrika*, **45**, 289–307.

[8] McArdle, J. J. and McDonald, R. P. (1984) Some algebraic properties of the reticular action model for moment structures. *British Journal of Mathematical and Statistical Psychology*, **37**, 234–251.

[9] Schlosser, R., Gesierich, T., Kaufmann, B., Vucurevic, G., Hunsche, S., Gawehn, J., and Stoeterb, P. (2003) Altered effective connectivity during working memory performance in schizophrenia: a study with fMRI and structural equation modeling. *NeuroImage*, **19**, 751–763.

[10] Au Duong, M., et al. (2005) Modulation of effective connectivity inside the working memory network in patients at the earliest stage of multiple sclerosis. *NeuroImage*, **24**, 533–538.

[11] Storkey, A. J., Simonotto, E., Whalley, H., Lawrie, S., Murray, L., and McGonigle, D. (2007) Learning structural equation models for fMRI. *Advances in Neural Information Processing Systems*, **19**.

[12] Buxton, R. B., Wong, E. C., and Frank, L. R. (1998) Dynamics of blood flow and oxygenation changes during brain activation: The balloon model. *Magnetic Resonance in Medicine*, **39**, 855–864.

[13] Friston, K. J., Mechelli, A., Turner, R., and Price, C. J. (2000) Nonlinear responses in fMRI: The balloon model, Volterra kernels, and other hemodynamics. *NeuroImage*, **12**, 466–477.

[14] Zarahn, E. (2001) Spatial localization and resolution of BOLD fMRI. *Current Opinion in Neurobiology*, **11**, 209–212.

[15] Isard, M. and Blake, A. (1998) Condensation – conditional density propagation for visual tracking. *International Journal of Computer Vision*, **29**, 5–28.

[16] Kitagawa, G. (1996) Monte Carlo filter and smoother for non-Gaussian nonlinear state space models. *Journal of Computational and Graphical Statistics*, **5**, 1–25.

[17] Isard, M. and Blake, A. (1998) A smoothing filter for condensation. *Proceedings of the 5th European Conference on Computer Vision*, **1**, 767–781.

[18] Kitagawa, G. (1998) A self-organising state-space model. *Journal of the American Statistical Association*, **93**, 1203–1215.

[19] Wellcome Department of Imaging Neuroscience (2006), Statistical parametric mapping. Online at www.fil.ion.ucl.ac.uk/spm/.

